# Revisiting Adversarial Patches for Designing Camera-Agnostic Attacks against Person Detection

Hui Wei[1*]      Zhixiang Wang[2*]      Kewei Zhang[1*]
Jiaqi Hou[1]      Yuanwei Liu[1]      Hao Tang[3]      Zheng Wang[1†]

[1]National Engineering Research Center for Multimedia Software,
School of Computer Science, Wuhan University
[2]The University of Tokyo      [3]School of Computer Science, Peking University
https://camera-agnostic.github.io/

## Abstract

Physical adversarial attacks can deceive deep neural networks (DNNs), leading to erroneous predictions in real-world scenarios. To uncover potential security risks, attacking the safety-critical task of person detection has garnered significant attention. However, we observe that existing attack methods overlook the pivotal role of the camera, involving capturing real-world scenes and converting them into digital images, in the physical adversarial attack workflow. This oversight leads to instability and challenges in reproducing these attacks. In this work, we revisit patch-based attacks against person detectors and introduce a camera-agnostic physical adversarial attack to mitigate this limitation. Specifically, we construct a differentiable camera Image Signal Processing (ISP) proxy network to compensate for the physical-to-digital transition gap. Furthermore, the camera ISP proxy network serves as a defense module, forming an adversarial optimization framework with the attack module. The attack module optimizes adversarial patches to maximize effectiveness, while the defense module optimizes the conditional parameters of the camera ISP proxy network to minimize attack effectiveness. These modules engage in an adversarial game, enhancing cross-camera stability. Experimental results demonstrate that our proposed Camera-Agnostic Patch (CAP) attack effectively conceals persons from detectors across various imaging hardware, including two distinct cameras and four smartphones.

## 1   Introduction

Adversarial attacks have emerged as a concerning threat to deep neural network (DNNs)-based models, casting a shadow over their reliability, particularly as certain attack methods extend beyond the digital space and prove effective in real-world scenarios [1, 5, 27]. Examples include wearing specialized glasses to mislead facial recognition models for impersonation attacks [26] or wearing clothing with adversarial textures to evade machine vision systems [14]. This category of attacks is commonly known as physical adversarial attacks [33].

Successfully executing physical adversarial attacks presents heightened challenges due to domain transitions and various dynamic physical factors encountered throughout the process from crafting digital perturbations to launching real-world attacks. Existing attack methods against person detectors have demonstrated notable advancements [32], and we categorize their efforts into two main types: **(1) Transitioning from the digital to the physical domain**, where techniques such as Non-Printability Scores (NPS) [26] are employed to mitigate color reproduction discrepancies caused

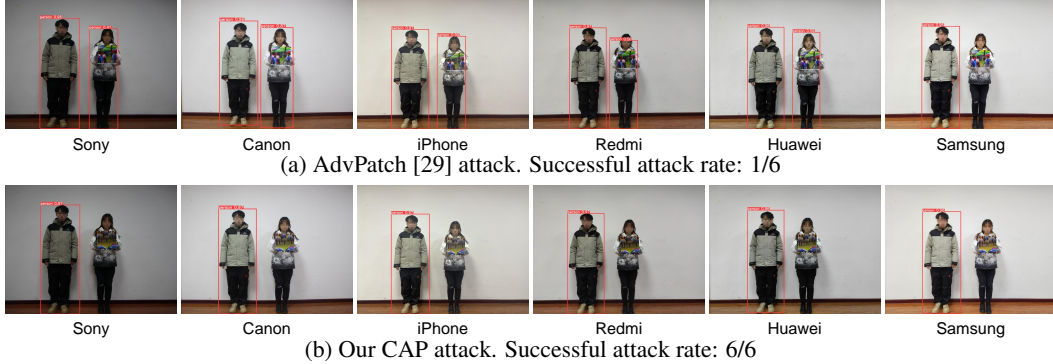

(a) AdvPatch [29] attack. Successful attack rate: 1/6

(b) Our CAP attack. Successful attack rate: 6/6

Figure 1: **Illustration depicting the impact of camera on attack performance.** The bounding boxes indicate the YOLOv5 [17] detector successfully detects the person. In each setting, we maintain scene consistency to minimize irrelevant influences. In contrast to the AdvPatch [29] attack, which is effective only on Samsung devices, our method successfully executes attacks across all six cameras.

by printers [29]. **(2) Transformations in the physical domain**, which involve using operations like rotations, scale variations, and others to simulate real-world variations [11, 28], leveraging Thin Plate Splines (TPS) to model cloth deformation [35], and utilizing fully-cover textures on clothing to handle multi-angle variations in the real world [13, 14]. Naturally, a question arises: Is it necessary to explore another transition, namely, **transitioning from the physical to the digital domain**?

In the journey from the physical scene to digital images, the camera plays a crucial role. This aspect has been overlooked for an extended period. Therefore, to shed light on the aforementioned question, we evaluated the camera's impact on attack performance. Specifically, we captured the same physical scene using different cameras (Sony, Canon, iPhone, *etc.*) and observed that the detection results for non-attack persons remained relatively stable, whereas, for persons with adversarial patches, the confidence values exhibited considerable variations. Some of the results are shown in Figure 1a. These experimental results demonstrate that the physical-to-digital domain transition, specifically the camera imaging pipeline's transformation of real-world scenes into digital counterparts, constitutes a crucial factor that significantly impacts adversarial attack performance.

Inspired by this observation, we are committed to designing *camera-agnostic* physical adversarial attacks. To maintain stable attack performance across a variety of imaging devices in the real world, our method introduces a camera simulation into the adversarial patch generation pipeline. Here we emphasize the significance of the camera ISP, a pivotal component that connects the RAW sensor data captured by the camera to the ultimate processed image. Our analysis reveals that camera ISP processing inherently attenuates attack performance, highlighting the camera ISP's potential defensive role against adversarial attacks, effectively positioning it as a natural defender. This observation aligns with Zhang *et al.* [39], who employed learned ISP pipelines to design an off-the-shelf preprocessing module for defending against digital adversarial attacks. Consequently, we propose an adversarial optimization framework to generate camera-agnostic adversarial patches. Specifically, a differentiable camera ISP proxy network functions as a defense module by adjusting conditional parameters to reduce the efficacy of adversarial patches. Conversely, the patch optimization module enhances attack performance by optimizing the patch itself. This adversarial optimization endows the generated patches with robust effectiveness across diverse camera hardware, as illustrated in Figure 1b.

In summary, our main contributions are as follows:

- A complete modeling of the workflow for physical adversarial attacks that integrates camera modules previously overlooked in existing research. Our method unveils the significant impact of the imaging devices and integrates a differentiable camera ISP proxy network into the attack pipeline.

- A new adversarial patch generation framework gains cross-camera attack capabilities. Our method leverages the camera ISP module's defense properties by optimizing conditional parameters to reduce patch effectiveness, establishing a zero-sum game with the perturbation

Table 1: **Summary of typical patch-based physical adversarial attacks against person detection.** While existing attack methods have partially addressed the Digital-to-Physical transition, none have systematically investigated the Physical-to-Digital transition. Our proposed CAP attack introduces a Camera Proxy Network to model this crucial transition and comprehensively evaluates attack performance across diverse unseen imaging devices.

| Categories | Method | Digital-to-Physical transition | Physical transformation | Physical-to-Digital transition | Black-box camera evaluation |
|---|---|---|---|---|---|
| Stealthiness | NAP (2021) [11] | ✗ | Scale, angle, *etc.* | ✗ | ✗ |
| | LAP (2021) [28] | NPS Loss | Scale, angle, *etc.* | ✗ | ✗ |
| Effectiveness & Robustness | AdvPatch (2019) [29] | NPS Loss | Scale, angle, *etc.* | ✗ | ✗ |
| | AdvT-shirt (2020) [35] | Color Transformer | TPS deformation | ✗ | ✗ |
| | AdvCloak (2020) [34] | Rendering Function | TPS deformation | ✗ | ✗ |
| | TC-EGA (2022) [14] | ✗ | TPS deformation | ✗ | ✗ |
| | T-SEA (2023) [15] | ✗ | Patch Cutout | ✗ | ✗ |
| | CAP (Ours) | NPS Loss | Scale, angle, *etc.* | Camera ISP Net | ✔ |

optimization module. This interaction ultimately strengthens the camera-agnostic robustness of the generated adversarial patch.

- Improved attack efficacy and heightened stability gains over existing methods. Real-world experiments demonstrate that our approach consistently and effectively achieves attack objectives across various imaging devices, including two typical cameras (Sony and Canon) and four smartphone cameras (iPhone, Redmi, Huawei, and Samsung).

## 2  Related Work

**Physical Adversarial Attacks on Vision Tasks**   Compared to digital adversarial attacks [2, 27, 36], physical adversarial attacks are more threatening because they can deceive DNNs-based models in the real world. Sharif *et al.* [26] achieved the first implementation of physical adversarial attacks, targeting facial recognition systems. Since then, researchers have been designing attacks for various computer vision tasks, including classification [1], detection [29], segmentation [22], depth estimation [3], and image captioning [38]. In general, these methods generate perturbations in the digital domain, then transform them into tangible physical entities, deploy them in real-world scenarios, capture them with cameras, and finally return to the digital domain to complete the attack[2] In this process, two domain transitions are experienced. The first, namely digital-to-physical, has been addressed by some works [16, 26]. However, the second, namely physical-to-digital, has been always overlooked, resulting in existing attack methods being unstable and difficult to reproduce. Our approach instead addresses this absence by incorporating a differentiable camera ISP network, thus constructing a more comprehensive perturbation generation pipeline.

**Adversarial Patches for Person Detection**   Due to the significance of human privacy and security, adversarial patches are widely employed for attacking person detection models in real-world scenarios [32]. We summarize recent work on patch-based physical adversarial attacks targeting person detection in Table 1. Although existing methods have made significant progress in terms of effectiveness [29, 34, 35], stealthiness [11, 28], and robustness [14, 15], they all overlook the widespread scenario of cross-camera attacks in the physical world. They assume a white-box camera system, which diminishes their effectiveness in real-world scenarios with unseen cameras. Therefore, we advocate for treating the camera system as a black box and propose a method for designing camera-agnostic adversarial patches.

**Camera Image Signal Processing Pipeline**   In the journey from the physical scene to digital RGB images, the camera's internal Image Signal Processing (ISP) pipeline plays a crucial role. The ISP pipeline is tasked with converting the RAW measurements captured by camera sensors into high-quality RGB images that are suitable for further analysis and human perception. It employs a range of techniques and algorithms, such as demosaicing [7, 18], denoising [9, 37], white balancing [10, 12], to enhance acquired data, mitigate noise artifacts, and correct for optical aberrations. Intuitively, Zhang

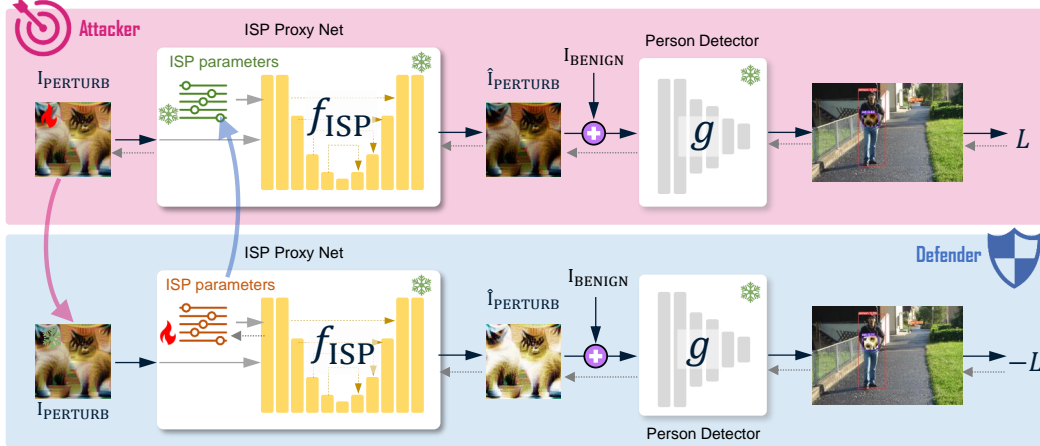

Figure 2: **Overview of our adversarial optimization framework.** The framework comprises two mutually adversarial parts: Attacker and Defender. The attacker optimizes adversarial perturbations to maximize attack effectiveness, while the defender optimizes the conditional input hyperparameter of the ISP proxy network to minimize attack effectiveness. The two parts cyclically alternate during the optimization stage.

*et al.* [39] discovered that the camera's ISP weakens the effectiveness of adversarial perturbations and developed an off-the-shelf preprocessing adversarial defense method. Inspired by this insight, our approach incorporates a differentiable camera ISP network as a defense module, designing an adversarial optimization framework to ensure attack robustness of generated adversarial perturbations across different camera ISP configurations.

## 3 Camera-Agnostic Attack

### 3.1 Problem Definition

A camera system plays a transformational role in converting the physical scene $I_{\mathrm{SCENE}}$ into its digital counterpart $I_{\mathrm{RGB}}$. Subsequently, the digital image $I_{\mathrm{RGB}}$ serves as input for well-trained downstream DNNs-based models $g$, producing predictions $y$ that closely align with the ground truth ($Y_{\mathrm{GT}}$). Our goal is to generate adversarial patches $P$ and apply them to $I_{\mathrm{SCENE}}$ to attack the model $g$ to cause incorrect predictions $y'$. Unlike existing camera-specific physical adversarial attack methods, our approach aims to maintain stable performance across various cameras. In our attack setting, we regard the imaging process (from $I_{\mathrm{SCENE}}$ to $I_{\mathrm{RGB}}$) as a black box.

### 3.2 Overall Framework

To enable the generated adversarial patches to adapt to various cameras, we introduce a novel adversarial optimization framework (see Figure 2). It consists of two mutually adversarial parts: Attacker and Defender. The attacker has an ISP proxy network on top of existing attacking strategies. The ISP network maps adversarial perturbations to the RGB space based on conditional input hyperparameters. The processed adversarial perturbations are subsequently applied to benign samples and fed into the target detection model to get predictions. The attacker iteratively optimized adversarial perturbations to deliberately deviate the person detector's output from the ground-truth labels through gradient descent [19]. The defender employs the same structure but different optimization strategy. It optimized the conditional hyperparameters to minimize the attack effectiveness of adversarial perturbations. During the attacker optimization phase, we freeze the conditional ISP hyperparameters, and similarly, during the defender optimization phase, we freeze the adversarial perturbations.

### 3.3 Differentiable Camera ISP Simulation

The camera ISP is responsible for converting the raw measurements of camera sensors into high-quality RGB images suitable for further analysis and human perception. It consists a range of

**Algorithm 1** The proposed adversarial optimization ( Attacker and Defender)

---
1: Given source image data $X$, targeted person detector $g$, and the trained camera ISP network $f_{\text{ISP}}$;
2: Initialize the adversarial patch $P$ and input hyperparameters $\boldsymbol{\Theta}$ of $f_{\text{ISP}}$;
3: **for** $t = 1, 2, \ldots, T$ **do**
4:      `// Optimize the adversarial patch `$P$` to maximize attack effectiveness`
5:      **for** batch $b = 1, 2, \ldots, M$ **do**
6:          Sample a batch of data $X_b$ from $X$;
7:          $X_{adv} \leftarrow \text{apply}(X_b, P_{\text{RGB}})$, $P_{\text{RGB}} = f_{\text{ISP}}(P, \boldsymbol{\Theta})$;
8:          $X_{adv}$ are fed into the person detector $g$ to obtain predictions and compute the loss $L$;
9:          Update the adversarial patch $P$ via Eq. 1;
10:      **end for**
11:      `// Optimize input hyperparameters `$\boldsymbol{\Theta}$` to minimize the attack effectiveness`
12:      **for** batch $b = 1, 2, \ldots, M$ **do**
13:          Sample a batch of data $x_b$ from $X$;
14:          $X_{adv} \leftarrow \text{apply}(x_b, P_{\text{RGB}})$, $P_{\text{RGB}} = f_{\text{ISP}}(P, \boldsymbol{\Theta})$;
15:          $X_{adv}$ are fed into the person detector $g$ to obtain predictions and compute the loss $L$;
16:          Update the input hyperparameters $\boldsymbol{\Theta}$ via Eq. 2;
17:      **end for**
18: **end for**

---

techniques and algorithms, such as demosaicing [7, 18], denoising [9, 37], white balancing [10, 12], to enhance acquired data, mitigate noise artifacts, and correct for optical aberrations.

Traditional ISPs are typically based on hand-crafted modules that are not differentiable [39]. Therefore, they are not able to be incorporated into the adversarial pattern design. We propose a differentiable camera ISP proxy $f_{\text{ISP}}$ that can simulate arbitrary parameterized configurations, which is inspired by the literature of ISP optimization [25, 31]. Specifically, we trained a variant of the U-Net CNN architecture [24] using data $\{I_{\text{SCENE}}, I_{\text{RGB}}, \boldsymbol{\Theta}\}$ obtained from traditional ISPs. Our network took the measurement $I_{\text{SCENE}}$ as the input, hyperparameters $\boldsymbol{\Theta}$ of the camera ISP as the condition, and was trained by minimize the reconstruction error between its prediction and $I_{\text{RGB}}$. The training utilized 2,270 data pairs generated by an open-source undifferentiable camera ISP simulator [23] and the COCO dataset [20].

Since the hyperparameters of a camera ISP can vary from one implementation to another, and are often specific to the hardware and software used in a particular camera system [31], we opt for representative parameters that have a *significant* impact on the final imaging and attack performance. We empirically select six parameters from the Color and Tone Correction module and the Denoising module in the camera ISP. To this end, we represent the camera ISP pipeline as a function $f_{\text{ISP}}$ parameterized by the conditional physical parameter $\boldsymbol{\Theta} = <a, b, \gamma, c, d, e>$. To enable the ISP proxy network to accommodate conditional input hyperparameters, we normalize the 6-dimensional hyperparameter to the $[0, 1]$ interval and concatenate them to the feature variables of the encoder.

### 3.4 Adversarial Optimization

Our objective is twofold: (1) to optimize adversarial perturbations for optimal attack effectiveness against the target neural network and (2) to optimize the input hyperparameters of the ISP proxy network for optimal defense effectiveness. Since two optimizations engage in a zero-sum game, we follow the same training strategy in the GAN framework [8] to simultaneously optimize both parameter sets. The optimization algorithm is illustrated in Algorithm 1. In practice, we employ iterative updates to implement alternating training. The process alternates between $k_1$ steps of optimizing the adversarial perturbation $P$ and $k_2$ steps of optimizing ISP conditional parameters ($<a, b, \gamma, c, d, e>$). We set $k_1 = k_2 = 20$. This strategy, validated through experiments, ensures the optimal optimization of both attack and defense, maintaining proximity to their peak values.

So, the goal can be described as follows:

$$P^* = \arg\max_i L(g(I_{\text{SCENE}}^i, f_{\text{ISP}}(P; \boldsymbol{\Theta})), Y_{\text{GT}}), \tag{1}$$

where we find optimal adversarial patches $P$ by maximizing the discrepancy $L$ between the predictions of the model $g$ and the ground truth $Y_{\text{GT}}$.

Additionally, we treat $f_{\text{ISP}}$ as a defense module, with the objective:

$$\boldsymbol{\Theta}^* = \arg\min_i L(g(I_{\text{SCENE}}^i, f_{\text{ISP}}(P; \boldsymbol{\Theta})), Y_{\text{GT}}), \tag{2}$$

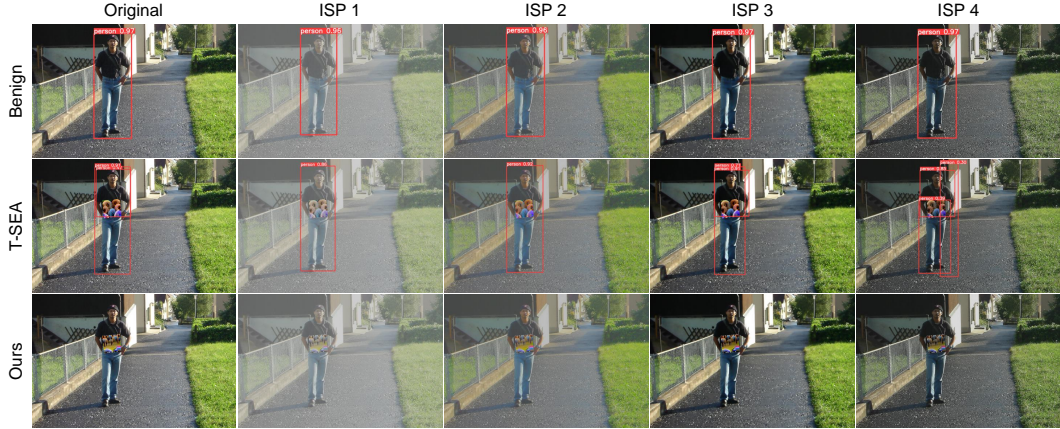

Figure 3: **Illustration of digital-space attacks under different ISP settings.** The bounding boxes indicate the detector successfully detects the person instances, *i.e.*, the attack fails. Due to space constraints, we only present one comparative method, T-SEA [15]. For additional results, please refer to the Supplementary Material.

where we find optimal conditional parameter $\Theta$ by minimizing the discrepancy $L$.

## 4 Experiments

### 4.1 Experimental Setup

**Datasets** We use the INRIAPERSON dataset [4, 30] to evaluate digital-space attacks. For physical-space attack, aiming to showcase the camera-agnostic nature of our approach, we collected data using six distinct hardware imaging devices, including two cameras — Sony $\alpha$7R4 and Canon DS126231 — and four mobile phone cameras — iPhone15, RedmiK20, HuaweiP50, and SamsungS22.

**Compared Methods** We compare our proposed method with seven mainstream patch-based methods, including AdvPatch [29], AdvT-shirt [35], AdvCloak [34], NAP [11], LAP [28], TC-EGA [14], and T-SEA [15]. For a fair comparison, we control the size of these patches to be the same, set at 0.2 times the height of the person.

**Metrics** We evaluate attack effectiveness using two primary metrics: Average Precision (AP%) and Attack Success Rate (ASR%). AP assesses detection model accuracy, where lower values indicate superior attack performance. ASR is defined as $1 - \mathrm{TP}'/\mathrm{TP}$, where TP denotes the number of True Positive detections without attacks and $\mathrm{TP}'$ represents those with attacks; higher ASR values indicate better attack performance.

For digital-space evaluation, we utilize the INRIAPerson dataset, which consists of 613 training images with 3,019 person instances and 288 test images containing 855 person instances. The ASR in the digital space is therefore calculated based on these 855 person instances across 288 test images. In the physical-space evaluation, we conducted data collection using 6 cameras across 4 temporal sessions to minimize confounding factors. For each patch configuration, we captured 5 images per camera per session, yielding 120 images (6×4×5) per patch. With 6 distinct adversarial patches evaluated in the physical domain, our analysis encompasses a total of 720 images, forming the basis for physical-space ASR calculations.

**Implementation Details** Our implementation utilizes PyTorch on a Linux server equipped with dual NVIDIA GeForce RTX 3090 GPUs. The adversarial patches are configured with dimensions of 300×300, and we employ a YOLOv5 [17] model pre-trained on the COCO dataset [20] and subsequently fine-tuned on INRIAPerson [30] as our victim detector. The detector processes input images at a resolution of 640×640, and adversarial training proceeds for 100 epochs.

Table 2: **Quantitative results of different attack methods under various ISP settings in digital space.** Our CAP attack surpasses all existing methods in terms of attack success rate (ASR%). The reason T-SEA [15] performs well in Average Precision (AP%) but poorly in ASR is due to the multiple bounding box detections. We discuss this conflict in Subsection 4.2.

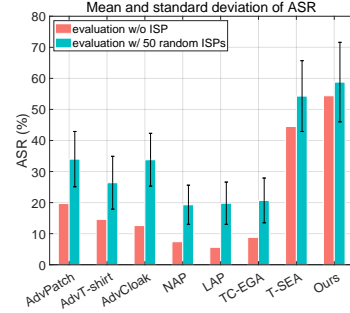

Figure 4: **Comparison of ASR (%)**. We evaluate various adversarial patches under 50 random camera ISPs in digital space.

| Method | Original | | ISP 1 | | ISP 2 | | ISP 3 | | ISP 4 | |
|---|---|---|---|---|---|---|---|---|---|---|
| | AP↓ | ASR↑ | AP↓ | ASR↑ | AP↓ | ASR↑ | AP↓ | ASR↑ | AP↓ | ASR↑ |
| Confidence threshold = 0.001, IoU threshold = 0.6 | | | | | | | | | | |
| Random Noise | 81.7 | 7.3 | 79.3 | 14.9 | 80.2 | 11.0 | 79.8 | 10.9 | 80.1 | 8.5 |
| AdvPatch [29] | 67.7 | 19.7 | 60.4 | 38.3 | 65.8 | 30.4 | 64.5 | 28.2 | 68.6 | 22.9 |
| AdvT-shirt [35] | 76.6 | 14.6 | 73.0 | 21.9 | 76.1 | 18.8 | 71.7 | 21.2 | 76.5 | 14.1 |
| AdvCloak [34] | 70.5 | 12.6 | 65.3 | 30.4 | 68.9 | 23.7 | 64.3 | 25.0 | 68.6 | 15.8 |
| NAP [11] | 81.3 | 7.4 | 76.8 | 16.9 | 79.1 | 12.9 | 76.5 | 13.8 | 80.2 | 8.8 |
| LAP [28] | 81.0 | 5.6 | 76.3 | 17.2 | 78.6 | 11.6 | 77.8 | 12.1 | 79.4 | 10.1 |
| TC-EGA [14] | 79.9 | 8.8 | 71.3 | 20.3 | 76.4 | 14.4 | 75.6 | 17.1 | 76.8 | 13.3 |
| T-SEA [15] | 21.2 | 44.5 | 27.0 | 53.0 | 22.8 | 52.7 | 26.3 | 44.7 | 24.7 | 47.4 |
| CAP (Ours) | 37.7 | 54.4 | 24.3 | 64.5 | 25.7 | 73.8 | 37.8 | 57.4 | 31.8 | 68.2 |

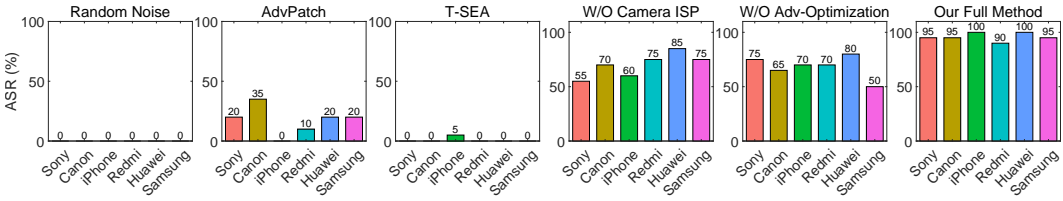

Figure 5: **Quantitative results of different attack methods in physical space.** For each adversarial patch, we evaluate its ASR (%) across six different cameras, including two typical cameras (Sony and Canon) and four smartphone cameras (iPhone, Redmi, Huawei, and Samsung).

## 4.2 Camera-Agnostic Attack in Digital Space

To intuitively demonstrate the impact of camera ISP on digital imaging and attack effectiveness, we selected 4 distinct camera ISP parameters for visualization (see Figure 3). We observe that different camera ISPs have little effect on the benign image, as the detector successfully detects person instances at all 4 camera ISP settings with little change in confidence. These results indicate that the detector is inherently robust, having camera-agnostic detection capabilities. The comparison method, T-SEA [15], failed to successfully attack all four camera ISPs. However, the confidence of person instances showed significant fluctuations. For instance, the confidence score of the person decreased from 0.92 (ISP 2) to 0.71 (ISP 3). In contrast, our method maintains stable attacks across all ISP settings, successfully concealing the person.

Table 2 reports the AP and ASR results of typical patch-based attack methods. We observe that the ASR of our CAP attack surpasses that of all comparison methods across all camera ISP settings. However, in terms of AP, T-SEA shows a more pronounced decrease compared to our method in most cases. Generally, a greater decrease in AP signifies a poorer detector performance, typically accompanied by a higher ASR. However, T-SEA deviates from this trend. To explore the reasons, we visualize the attack results of T-SEA and find a severe *multi-box detection* issue (refer to Supplementary Material Section C). A significant number of False Positive samples contribute to the decrease in AP, rather than effectively concealing person instances.

Furthermore, we present a comparison of the ASR for each adversarial patch in two settings: without camera ISP and with 50 random camera ISPs as preprocessing (see Figure 4). This comparison illustrates that our approach mitigates the instability of cross-camera attacks and enhances the attack efficacy of adversarial patches.

## 4.3 Camera-Agnostic Attack in Physical Space

Our CAP attack is designed for real-world scenarios where the target system's camera is unknown. Therefore, we compare the physical-space attack performance of different adversarial patches across six hardware cameras, including two typical cameras (Sony and Canon) and four smartphone cameras

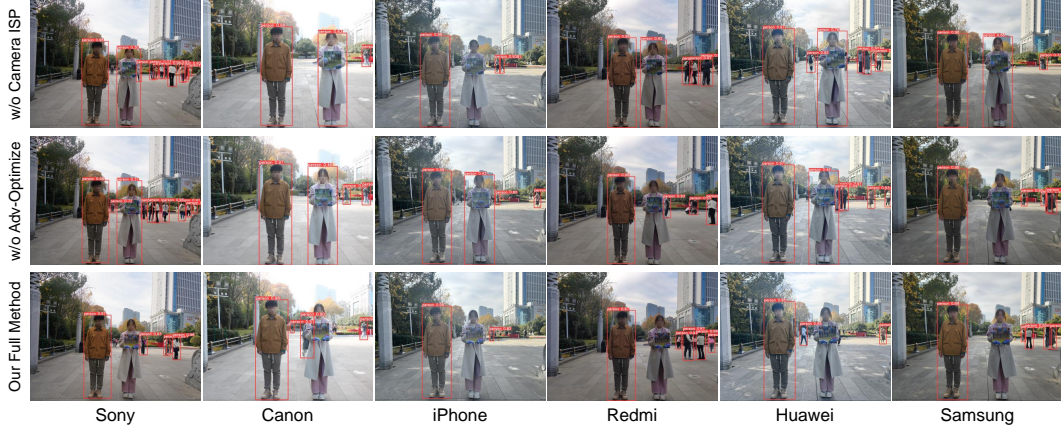

Figure 6: **Physical-space attacks across six different cameras.** Our method removing the camera ISP module only achieves successful attacks on specific cameras. Our method removing the adversarial optimization slightly outperforms the former. In contrast, our full method achieves successful attacks across all six cameras.

(iPhone, Redmi, Huawei, and Samsung). In the capture scenario, one participant carried various adversarial patches, while another participant, serving as the control, did not carry any adversarial patches (see Figure 1 and Figure 6). To eliminate interference, we captured 20 images for each adversarial patch setting and calculated the ASR (%). For further demonstrations of physical-space attacks, please refer to the Supplementary Material. Here, we only evaluate two comparison methods, AdvPatch [29] and T-SEA [15], since they primarily target attack performance (as evident from Table 2), unlike other methods [11, 28] that also consider stealthiness.

Figure 3 presents the quantitative results of the effectiveness of the attack for different adversarial patches. We observe that random noise-based patches exhibit no attack effectiveness in real-world scenarios. The attack performance of AdvPatch exhibits significant fluctuations. It achieves an ASR of 35% on Canon cameras, while it is 0% on iPhone cameras. Unlike its impressive performance in the digital space, T-SEA shows poor attack performance in the physical space, mostly unable to execute successful attacks. This is due to the *multi-box detection* issue. When computing ASR, we consider a sample as a failed attack if it is detected, even if the detection bounding box only covers half of the complete body. Our method achieves an ASR of more than 90% in all cameras, reaching 100% ASR on the iPhone and Huawei. These results demonstrate the excellent camera-agnostic attack performance of our method in physical space.

## 4.4 Ablation Study

To demonstrate the effectiveness of each component in our attack method, we perform two variants of our method, *i.e.* ours w/o the camera ISP module and ours w/o adversarial optimization, as shown in Figure 6. Note that the latter refers to retaining the camera ISP module in our pipeline but refraining from optimizing its conditional input parameters. Instead, during the perturbation optimization process, we randomly adjust the input parameters.

From Figure 6, we observe that (1) "ours w/o camera ISP" exhibits the greatest fluctuation. This is evident from the confidence scores of the persons with the adversarial patches. It reaches as high as 0.92 on Redmi devices, while it drops below 0.25 on iPhone devices (indicating the disappearance of detection boxes). (2) The attack effectiveness of "ours w/o adversarial optimization" surpasses that of the former group. It succeeds in attacking both the Redmi and Samsung devices, with a noticeable reduction in the confidence fluctuations of the victim person. (3) Our full method achieves successful attacks across all six cameras. Additionally, Figure 5 presents the quantitative comparison of ASR across different cameras for three settings. Compared to the two variants, our method achieves a higher and more stable ASR. These results demonstrate that incorporating the camera ISP module solely into the adversarial perturbation pipeline offers limited improvement in attack performance, while our proposed adversarial optimization design enhances cross-camera attack capability and stability in the real world.

Table 3: **Defenses against CAP attacks.** We report the AP in digital space and the ASR in physical space for three defense strategies: JPEG compression [6], SAC [21], and adversarial training [40].

(a) AP (%) in digital space

| Defense strategy \ Attack method | Non-attack | CAP* | CAP† | CAP |
|---|---|---|---|---|
| Non-defense | 85.0 | 52.8 | 45.5 | 37.7 |
| JPEG compression [6] | 84.7 | 52.7 | 45.8 | 36.8 |
| SAC [21] | 85.0 | 56.2 | 52.2 | 46.0 |
| Adversarial training-CAP* | 84.1 | 95.7 | 91.7 | 94.3 |
| Adversarial training-CAP† | 84.0 | 92.6 | 95.4 | 92.8 |
| Adversarial training-CAP | 84.6 | 94.2 | 91.6 | 96.3 |

(b) ASR (%) in physical space

| Defense strategy \ Attack method | CAP* | CAP† | CAP |
|---|---|---|---|
| Non-defense | 70.0 | 68.3 | 95.8 |
| JPEG compression [6] | 90.8 | 89.2 | 93.3 |
| SAC [21] | 70.0 | 68.3 | 95.8 |
| Adversarial training-CAP* | 0.0 | 4.2 | 1.7 |
| Adversarial training-CAP† | 5.8 | 0.0 | 10.8 |
| Adversarial training-CAP | 0.0 | 4.0 | 0.0 |

## 5 Discussions

**Defense** We compared three types of defense methods against CAP attacks: (1) modifying input images using JPEG compression [6], (2) adversarial patch detection and removal via SAC [21], and (3) adversarial training [40]. To understand the effectiveness of existing defenses against our attack, we evaluated our method and its two variants under three defense strategies. CAP* refers to our method without the camera ISP module, CAP† refers to our method without adversarial optimization, and CAP refers to our full method. In Table 3, we report the AP in the digital space and the ASR in the physical space for each case.

Overall, we observe that JPEG compression is ineffective against all three attack settings. This indicates that minor pixel-level modifications cannot defend against our CAP attacks. SAC demonstrates some defensive capability in digital space, slightly increasing the AP, but it is ineffective in physical space. In contrast, adversarial training effectively defends against CAP attacks with minimal loss in detector accuracy (within 1%). Furthermore, adversarial training shows defensive transferability in all three attack settings. One of the primary objectives of our study is to enhance the robustness of person detection models. The above results indicate that adversarial training is a reasonable and effective method to improve the robustness of detectors against CAP attacks.

**Limitations** Our study mainly focuses on utilizing a camera ISP proxy network for camera simulation, handling the transition from physical to digital domains. Building a comprehensive, end-to-end differentiable camera simulator that includes features such as exposure time, aperture size, and ISO is challenging. Despite this, we believe that the conclusions and insights of this work are generalizable. This study successfully exposes previous methodological flaws and emphasizes the importance of considering the camera as a crucial module in the workflow of physical adversarial attacks.

**Ethics Statement** Our work successfully achieves physical adversarial attacks in person detection tasks. Given the effectiveness of our attack method across various imaging devices, its real-world application is feasible. This exposes potential security risks in existing DNNs-based applications, particularly when the technology is leveraged for malicious purposes. We advocate for the responsible and ethical use of technology. Furthermore, we offer comprehensive methodological descriptions and openly address the implications of our work, encouraging discourse within and beyond the scientific community to contribute to the advancement of trustworthy and dependable AI.

## 6 Conclusion

In this paper, we have proposed a cross-camera physical adversarial attack, CAP (Camera-Agnostic Patch) attack, against person detection. Unlike previous methods that overlooked the crucial role of the camera in the real-world attack workflow, our method incorporates a differentiable camera Image Signal Processing (ISP) proxy network to compensate for the physical-to-digital transition gap. Furthermore, leveraging the attenuating effect of camera ISP on attack performance, we construct an adversarial optimization framework. In this framework, the attack module optimizes adversarial perturbations, aiming to maximize attack effectiveness, while the defense module optimizes the input parameters conditionally, aiming to minimize attack effectiveness. The two modules alternate optimization, encouraging the generated adversarial patches to exhibit stability across camera attacks.

Extensive experiments conducted in both digital and physical spaces demonstrate that our CAP attack enhances the effectiveness and reliability in real-world scenarios, encountering diverse camera configurations. In the future, we will continue to explore the role of cameras, design defense strategies based on imaging devices, and develop more robust detection models.

## Acknowledgements

This work was supported by Hubei Key R&D Project (2022BAA033), National Natural Science Foundation of China (62171325).

## Footnotes

*Equal contribution      †Corresponding author

[2]Refer to Supplementary Material Figure A for an illustration of the workflow of physical adversarial attacks.

## References

[1] Tom B Brown, Dandelion Mané, Aurko Roy, Martín Abadi, and Justin Gilmer. Adversarial patch. In *Proceedings of the Advances in Neural Information Processing Systems Workshop*, 2017.

[2] Zikui Cai, Xinxin Xie, Shasha Li, Mingjun Yin, Chengyu Song, Srikanth V Krishnamurthy, Amit K Roy-Chowdhury, and M Salman Asif. Context-aware transfer attacks for object detection. In *AAAI Conference on Artificial Intelligence*, 2022.

[3] Zhiyuan Cheng, James Liang, Hongjun Choi, Guanhong Tao, Zhiwen Cao, Dongfang Liu, and Xiangyu Zhang. Physical attack on monocular depth estimation with optimal adversarial patches. In *European Conference on Computer Vision*, pages 514–532. Springer, 2022.

[4] Navneet Dalal and Bill Triggs. Histograms of oriented gradients for human detection. In *Proceedings of the IEEE/CVF Conference on Computer Vision and Pattern Recognition*, volume 1, pages 886–893. IEEE, 2005.

[5] Yinpeng Dong, Shouwei Ruan, Hang Su, Caixin Kang, Xingxing Wei, and Jun Zhu. Viewfool: Evaluating the robustness of visual recognition to adversarial viewpoints. *Advances in Neural Information Processing Systems*, 35:36789–36803, 2022.

[6] Gintare Karolina Dziugaite, Zoubin Ghahramani, and Daniel M Roy. A study of the effect of jpg compression on adversarial images. *arXiv preprint arXiv:1608.00853*, 2016.

[7] Michaël Gharbi, Gaurav Chaurasia, Sylvain Paris, and Frédo Durand. Deep joint demosaicking and denoising. *ACM Transactions on Graphics (ToG)*, 35(6):1–12, 2016.

[8] Ian Goodfellow, Jean Pouget-Abadie, Mehdi Mirza, Bing Xu, David Warde-Farley, Sherjil Ozair, Aaron Courville, and Yoshua Bengio. Generative adversarial nets. *Advances in neural information processing systems*, 27, 2014.

[9] Shi Guo, Zifei Yan, Kai Zhang, Wangmeng Zuo, and Lei Zhang. Toward convolutional blind denoising of real photographs. In *Proceedings of the IEEE/CVF conference on computer vision and pattern recognition*, pages 1712–1722, 2019.

[10] Eugene Hsu, Tom Mertens, Sylvain Paris, Shai Avidan, and Frédo Durand. Light mixture estimation for spatially varying white balance. In *ACM SIGGRAPH 2008 papers*, pages 1–7. 2008.

[11] Yu-Chih-Tuan Hu, Bo-Han Kung, Daniel Stanley Tan, Jun-Cheng Chen, Kai-Lung Hua, and Wen-Huang Cheng. Naturalistic physical adversarial patch for object detectors. In *IEEE/CVF International Conference on Computer Vision*, 2021.

[12] Yuanming Hu, Baoyuan Wang, and Stephen Lin. Fc4: Fully convolutional color constancy with confidence-weighted pooling. In *Proceedings of the IEEE conference on computer vision and pattern recognition*, pages 4085–4094, 2017.

[13] Zhanhao Hu, Wenda Chu, Xiaopei Zhu, Hui Zhang, Bo Zhang, and Xiaolin Hu. Physically realizable natural-looking clothing textures evade person detectors via 3d modeling. In *Proceedings of the IEEE/CVF Conference on Computer Vision and Pattern Recognition*, pages 16975–16984, 2023.

[14] Zhanhao Hu, Siyuan Huang, Xiaopei Zhu, Fuchun Sun, Bo Zhang, and Xiaolin Hu. Adversarial texture for fooling person detectors in the physical world. In *Proceedings of the IEEE/CVF Conference on Computer Vision and Pattern Recognition*, pages 13307–13316, 2022.

[15] Hao Huang, Ziyan Chen, Huanran Chen, Yongtao Wang, and Kevin Zhang. T-sea: Transfer-based self-ensemble attack on object detection. In *Proceedings of the IEEE/CVF Conference on Computer Vision and Pattern Recognition*, pages 20514–20523, 2023.

[16] Steve TK Jan, Joseph Messou, Yen-Chen Lin, Jia-Bin Huang, and Gang Wang. Connecting the digital and physical world: Improving the robustness of adversarial attacks. In *Proceedings of the AAAI Conference on Artificial Intelligence*, volume 33, pages 962–969, 2019.

[17] Glenn Jocher. Ultralytics yolov5, 2020.

[18] Daniel Khashabi, Sebastian Nowozin, Jeremy Jancsary, and Andrew W Fitzgibbon. Joint demosaicing and denoising via learned nonparametric random fields. *IEEE Transactions on Image Processing*, 23(12):4968–4981, 2014.

[19] Diederik P. Kingma and Jimmy Ba. Adam: A method for stochastic optimization. In *3rd International Conference on Learning Representations, ICLR 2015, San Diego, CA, USA, May 7-9, 2015, Conference Track Proceedings*, 2015.

[20] Tsung-Yi Lin, Michael Maire, Serge Belongie, James Hays, Pietro Perona, Deva Ramanan, Piotr Dollár, and C Lawrence Zitnick. Microsoft coco: Common objects in context. In *European conference on computer vision*, 2014.

[21] Jiang Liu, Alexander Levine, Chun Pong Lau, Rama Chellappa, and Soheil Feizi. Segment and complete: Defending object detectors against adversarial patch attacks with robust patch detection. In *Proceedings of the IEEE/CVF Conference on Computer Vision and Pattern Recognition*, pages 14973–14982, 2022.

[22] Federico Nesti, Giulio Rossolini, Saasha Nair, Alessandro Biondi, and Giorgio Buttazzo. Evaluating the robustness of semantic segmentation for autonomous driving against real-world adversarial patch attacks. In *Proceedings of the IEEE/CVF Winter Conference on Applications of Computer Vision*, pages 2280–2289, 2022.

[23] Jueqin Qiu. Fast open image signal processor (fast-openisp). `https://github.com/QiuJueqin/fast-openISP`, 2021.

[24] Olaf Ronneberger, Philipp Fischer, and Thomas Brox. U-net: Convolutional networks for biomedical image segmentation. In *Medical Image Computing and Computer-Assisted Intervention–MICCAI 2015: 18th International Conference, Munich, Germany, October 5-9, 2015, Proceedings, Part III 18*, pages 234–241. Springer, 2015.

[25] Eli Schwartz, Raja Giryes, and Alex M Bronstein. Deepisp: Toward learning an end-to-end image processing pipeline. *IEEE Transactions on Image Processing*, 28(2):912–923, 2018.

[26] Mahmood Sharif, Sruti Bhagavatula, Lujo Bauer, and Michael K Reiter. Accessorize to a crime: Real and stealthy attacks on state-of-the-art face recognition. In *Proceedings of the 2016 acm sigsac conference on computer and communications security*, pages 1528–1540, 2016.

[27] Christian Szegedy, Wojciech Zaremba, Ilya Sutskever, Joan Bruna, Dumitru Erhan, Ian J. Goodfellow, and Rob Fergus. Intriguing properties of neural networks. In Yoshua Bengio and Yann LeCun, editors, *2nd International Conference on Learning Representations, ICLR 2014, Banff, AB, Canada, April 14-16, 2014, Conference Track Proceedings*, 2014.

[28] Jia Tan, Nan Ji, Haidong Xie, and Xueshuang Xiang. Legitimate adversarial patches: Evading human eyes and detection models in the physical world. In *ACM International Conference on Multimedia*, 2021.

[29] Simen Thys, Wiebe Van Ranst, and Toon Goedemé. Fooling automated surveillance cameras: adversarial patches to attack person detection. In *Proceedings of the IEEE/CVF conference on computer vision and pattern recognition workshops*, pages 0–0, 2019.

[30] Pascal to Yolo. Inria person detection dataset dataset. `https://universe.roboflow.com/pascal-to-yolo-8yygq/inria-person-detection-dataset`, dec 2022. visited on 2023-10-28.

[31] Ethan Tseng, Felix Yu, Yuting Yang, Fahim Mannan, Karl ST Arnaud, Derek Nowrouzezahrai, Jean-François Lalonde, and Felix Heide. Hyperparameter optimization in black-box image processing using differentiable proxies. *ACM Trans. Graph.*, 38(4):27–1, 2019.

[32] Hui Wei, Hao Tang, Xuemei Jia, Zhixiang Wang, Hanxun Yu, Zhubo Li, Shin'ichi Satoh, Luc Van Gool, and Zheng Wang. Physical adversarial attack meets computer vision: A decade survey. *IEEE Transactions on Pattern Analysis and Machine Intelligence*, 2024.

[33] Xingxing Wei, Bangzheng Pu, Jiefan Lu, and Baoyuan Wu. Visually adversarial attacks and defenses in the physical world: A survey. *arXiv preprint arXiv:2211.01671*, 2022.

[34] Zuxuan Wu, Ser-Nam Lim, Larry S Davis, and Tom Goldstein. Making an invisibility cloak: Real world adversarial attacks on object detectors. In *European Conference on Computer Vision*, 2020.

[35] Kaidi Xu, Gaoyuan Zhang, Sijia Liu, Quanfu Fan, Mengshu Sun, Hongge Chen, Pin-Yu Chen, Yanzhi Wang, and Xue Lin. Adversarial t-shirt! evading person detectors in a physical world. In *European conference on computer vision*, 2020.

[36] Qiuling Xu, Guanhong Tao, Siyuan Cheng, and Xiangyu Zhang. Towards feature space adversarial attack by style perturbation. In *AAAI Conference on Artificial Intelligence*, volume 35, pages 10523–10531, 2021.

[37] Kai Zhang, Wangmeng Zuo, and Lei Zhang. Ffdnet: Toward a fast and flexible solution for cnn-based image denoising. *IEEE Transactions on Image Processing*, 27(9):4608–4622, 2018.

[38] Shibo Zhang, Yushi Cheng, Wenjun Zhu, Xiaoyu Ji, and Wenyuan Xu. Capatch: Physical adversarial patch against image captioning systems. 2023.

[39] Yuxuan Zhang, Bo Dong, and Felix Heide. All you need is raw: Defending against adversarial attacks with camera image pipelines. In *European Conference on Computer Vision*, pages 323–343. Springer, 2022.

[40] Dawei Zhou, Nannan Wang, Bo Han, and Tongliang Liu. Modeling adversarial noise for adversarial training. In *International Conference on Machine Learning*, pages 27353–27366. PMLR, 2022.

